# New Approximations of Differential Entropy for Independent Component Analysis and Projection Pursuit

**Aapo Hyvärinen**
Helsinki University of Technology
Laboratory of Computer and Information Science
P.O. Box 2200, FIN-02015 HUT, Finland
Email: aapo.hyvarinen@hut.fi

## Abstract

We derive a first-order approximation of the density of maximum entropy for a continuous 1-D random variable, given a number of simple constraints. This results in a density expansion which is somewhat similar to the classical polynomial density expansions by Gram-Charlier and Edgeworth. Using this approximation of density, an approximation of 1-D differential entropy is derived. The approximation of entropy is both more exact and more robust against outliers than the classical approximation based on the polynomial density expansions, without being computationally more expensive. The approximation has applications, for example, in independent component analysis and projection pursuit.

## 1  Introduction

The basic information-theoretic quantity for continuous one-dimensional random variables is differential entropy. The differential entropy $H$ of a scalar random variable $X$ with density $f(x)$ is defined as

$$H(X) = -\int f(x) \log f(x) dx. \tag{1}$$

The 1-D differential entropy, henceforth called simply entropy, has important applications such areas as independent component analysis [2, 10] and projection pursuit [5, 6]. Indeed, both of these methods can be considered as a search for directions in which entropy is minimal, for constant variance.

Unfortunately, the estimation of entropy is quite difficult in practice. Using definition (1) requires estimation of the density of $X$, which is recognized to be both

theoretically difficult and computationally demanding. Simpler approximations of entropy have been proposed both in the context of projection pursuit [9] and independent component analysis [1, 2]. These approximations are usually based on approximating the density $f(x)$ using the polynomial expansions of Gram-Charlier or Edgeworth [11]. This construction leads to the use of higher-order cumulants, like kurtosis. However, such cumulant-based methods often provide a rather poor approximation of entropy. There are two main reasons for this. Firstly, finite-sample estimators of higher-order cumulants are highly sensitive to outliers: their values may depend on only a few, possibly erroneous, observations with large values [6]. This means that outliers may completely determine the estimates of cumulants, thus making them useless. Secondly, even if the cumulants were estimated perfectly, they measure mainly the tails of the distribution, and are largely unaffected by structure near the centre of the distribution [5].

Therefore, better approximations of entropy are needed. To this end, we introduce in this paper approximations of entropy that are both more exact in the expectation and have better finite-sample statistical properties, when compared to the cumulant-based approximations. Nevertheless, they retain the computational and conceptual simplicity of the cumulant-based approach. Our approximations are based on an approximative maximum entropy method. This means that we approximate the *maximum* entropy that is compatible with our measurements of the random variable $X$. This maximum entropy, or further approximations thereof, can then be used as a meaningful approximation of the entropy of $X$. To accomplish this, we derive a first-order approximation of the density that has the maximum entropy given a set of constraints, and then use it to derive approximations of the differential entropy of $X$.

## 2 Applications of Differential Entropy

First, we discuss some applications of the approximations introduced in this paper. Two important applications of differential entropy are independent component analysis (ICA) and projection pursuit. In the general formulation of ICA [2], the purpose is to transform an observed random vector $\mathbf{x} = (x_1, ..., x_m)^T$ linearly into a random vector $\mathbf{s} = (s_1, ..., s_m)^T$ whose components are statistically as independent from each other as possible. The mutual dependence of the $s_i$ is classically measured by mutual information. Assuming that the linear transformation is invertible, the mutual information $I(s_1, ..., s_m)$ can be expressed as $I(s_1, ..., s_m) = \sum_i H(s_i) - H(x_1, ..., x_m) - \log |\det \mathbf{M}|$ where $\mathbf{M}$ is the matrix defining the transformation $\mathbf{s} = \mathbf{Mx}$. The second term on the right-hand side does not depend on $\mathbf{M}$, and the minimization of the last term is a simple matter of differential calculus. Therefore, the critical part is the estimation of the 1-D entropies $H(s_i)$: finding an efficient and reliable estimator or approximation of entropy enables an efficient and reliable estimation of the ICA decomposition.

In projection pursuit, the purpose is to search for projections of multivariate data which have 'interesting' distributions [5, 6, 9]. Typically, interestingness is considered equivalent with non-Gaussianity. A natural criterion of non-Gaussianity is entropy [6, 9], which attains its maximum (for constant variance) when the distribution is Gaussian, and all other distributions have smaller entropies. Because of the difficulties encountered in the estimation of entropy, many authors have considered other measures of non-Gaussianity (see [3]) but entropy remains, in our view, the best choice of a projection pursuit index, especially because it provides a simple connection to ICA. Indeed, it can be shown [2] that in ICA as well as in projection pursuit, the basic problem is to find directions in which entropy is minimized for

constant variance.

## 3 Why maximum entropy?

Assume that the information available on the density $f(x)$ of the scalar random variable $X$ is of the form

$$\int f(x)G_i(x)dx = c_i, \text{ for } i = 1, ..., n, \tag{2}$$

which means in practice that we have estimated the expectations $E\{G_i(X)\}$ of $n$ different functions of $X$. Since we are not assuming any model for the random variable $X$, the estimation of the entropy of $X$ using this information is not a well-defined problem: there exist an infinite number of distributions for which the constraints in (2) are fulfilled, but whose entropies are very different from each other. In particular, the differential entropy reaches $-\infty$ in the limit where $X$ takes only a finite number of values.

A simple solution to this dilemma is the maximum entropy method. This means that we compute the *maximum* entropy that is compatible with our constraints or measurements in (2), which is a well-defined problem. This maximum entropy, or further approximations thereof, can then be used as an approximation of the entropy of $X$.

Our approach thus is very different from the asymptotic approach often used in projection pursuit [3, 5]. In the asymptotic approach, one establishes a sequence of functions $G_i$ so that when $n$ goes to infinity, the information in (2) gives an asymptotically convergent approximation of some theoretical projection pursuit index. We avoid in this paper any asymptotic considerations, and consider directly the case of finite information, i.e., finite $n$. This non-asymptotic approach is justified by the fact that often in practice, only a small number of measurements of the form (2) are used, for computational or other reasons.

## 4 Approximating the maximum entropy density

In this section, we shall derive an approximation of the density of maximum entropy compatible with the measurements in (2). The basic results of the maximum entropy method tell us [4] that under some regularity conditions, the density $f_0(x)$ which satisfies the constraints (2) and has maximum entropy among all such densities, is of the form

$$f_0(x) = A \exp(\sum_i a_i G_i(x)), \tag{3}$$

where $A$ and $a_i$ are constants that are determined from the $c_i$, using the constraints in (2) (i.e., by substituting the right-hand side of (3) for $f$ in (2)), and the constraint $\int f_0(x)dx = 1$. This leads in general to a system of $n+1$ non-linear equations which is difficult to solve. Therefore, we decide to make a simple approximation of $f_0$. This is based on the assumption that the density $f(x)$ is *not very far from a Gaussian distribution* of the same mean and variance. Such an assumption, though perhaps counterintuitive, is justified because we shall construct a density expansion (not unlike a Taylor expansion) in the vicinity of the Gaussian density. In addition, we can make the technical assumption that $f(x)$ is near the standardized Gaussian density $\varphi(x) = \exp(-x^2/2)/\sqrt{2\pi}$, since this amounts simply to making $X$ zero-mean and of unit variance. Therefore we put two additional constraints in (2), defined by $G_{n+1}(x) = x, c_{n+1} = 0$ and $G_{n+2}(x) = x^2, c_{n+2} = 1$. To further simplify

the calculations, let us make another, purely technical assumption: The functions $G_i, i = 1, ..., n$, form an orthonormal system according to the metric defined by $\varphi$, and are orthogonal to all polynomials of second degree. In other words, for all $i, j = 1, ..., n$

$$\int \varphi(x)G_i(x)G_j(x)dx = \left\{ \begin{array}{l} 1, \text{ if } i = j \\ 0, \text{ if } i \neq j \end{array} \right. , \quad \int \varphi(x)G_i(x)x^k dx = 0, k = 0, 1, 2. \quad (4)$$

For any linearly independent functions $G_i$, this assumption can always be made true by ordinary Gram-Schmidt orthonormalization.

Now, note that the assumption of near-Gaussianity implies that all the other $a_i$ in (3) are very small compared to $a_{n+2} \approx -1/2$, since the exponential in (3) is not far from $\exp(-x^2/2)$. Thus we can make a first-order approximation of the exponential function (detailed derivations can be found in [8]). This allows for simple solutions for the constants in (3), and we obtain the *approximative maximum entropy density*, which we denote by $\hat{f}(x)$:

$$\hat{f}(x) = \varphi(x)(1 + \sum_{i=1}^{n} c_i G_i(x)) \quad (5)$$

where $c_i = E\{G_i(X)\}$. To estimate this density in practice, the $c_i$ are estimated, for example, as the corresponding sample averages of the $G_i(X)$. The density expansion in (5) is somewhat similar to the Gram-Charlier and Edgeworth expansions [11].

## 5   Approximating the differential entropy

An important application of the approximation of density shown in (5) is in approximation of entropy. A simple approximation of entropy can be found by approximating both occurences of $f$ in the definition (1) by $\hat{f}$ as defined in Eq. (5), and using a Taylor approximation of the logarithmic function, which yields $(1 + \epsilon)\log(1 + \epsilon) \approx \epsilon + \epsilon^2/2$. Thus one obtains after some algebraic manipulations [8]

$$H(X) \approx -\int \hat{f}(x) \log \hat{f}(x)dx \approx H(\nu) - \frac{1}{2}\sum_{i=1}^{n} c_i^2 \quad (6)$$

where $H(\nu) = \frac{1}{2}(1 + \log(2\pi))$ means the entropy of a standardized Gaussian variable, and $c_i = E\{G_i(X)\}$ as above. Note that even in cases where this approximation is not very accurate, (6) can be used to construct a projection pursuit index (or a measure of non-Gaussianity) that is consistent in the sense that (6) obtains its maximum value, $H(\nu)$, when $X$ has a Gaussian distribution.

## 6   Choosing the measuring functions

Now it remains to choose the 'measuring' functions $G_i$ that define the information given in (2). As noted in Section 4, one can take practically any set of linearly independent functions, say $\bar{G}_i, i = 1, ..., n$, and then apply Gram-Schmidt orthonormalization on the set containing those functions and the monomials $x^k, k = 0, 1, 2$, so as to obtain the set $G_i$ that fulfills the orthogonality assumptions in (4). This can be done, in general, by numerical integration. In the practical choice of the functions $\bar{G}_i$, the following criteria must be emphasized: First, the practical estimation of $E\{\bar{G}_i(x)\}$ should not be statistically difficult. In particular, this estimation should not be too sensitive to outliers. Second, the maximum entropy method assumes

that the function $f_0$ in (3) is integrable. Therefore, to ensure that the maximum entropy distribution exists in the first place, the $\bar{G}_i(x)$ must not grow faster than quadratically as a function of $|x|$, because a function growing faster might lead to non-integrability of $f_0$ [4]. Finally, the $\bar{G}_i$ must capture aspects of the distribution of $X$ that are pertinent in the computation of entropy. In particular, if the density $f(x)$ were known, the optimal function $\bar{G}_{opt}$ would clearly be $-\log f(x)$, because $-E\{\log f(X)\}$ gives directly the entropy. Thus, one might use the log-densities of some known important densities as $\bar{G}_i$.

The first two criteria are met if the $\bar{G}_i(x)$ are functions that do not grow too fast (not faster than quadratically) when $|x|$ grows. This excludes, for example, the use of higher-order polynomials, as are used in the Gram-Charlier and Edgeworth expansions. One might then search, according to the last criterion above, for log-densities of some well-known distributions that also fulfill the first two conditions. Examples will be given in the next section. It should be noted, however, that the criteria above only delimit the space of function that can be used. Our framework enables the use of very different functions (or just one) as $\bar{G}_i$. The choice is not restricted to some well-known basis of a functional space, as in most approaches [1, 2, 9]. However, if prior knowledge is available on the distributions whose entropy is to estimated, the above consideration shows how to choose the optimal function.

## 7  A simple special case

A simple special case of (5) is obtained if one uses two functions $\bar{G}_1$ and $\bar{G}_2$, which are chosen so that $\bar{G}_1$ is *odd* and $\bar{G}_2$ is *even*. Such a system of two functions can measure the two most important features of non-Gaussian 1-D distributions. The odd function measures the asymmetry, and the even function measures the bimodality/sparsity dimension (called central hole/central mass concentration in [3]). After extensive experiments, Cook et al [3] also came to the conclusion that two such measures (or two terms in their projection pursuit index) are enough for projection pursuit in most cases. Classically, these features have been measured by skewness and kurtosis, which correspond to $\bar{G}_1(x) = x^3$ and $\bar{G}_2(x) = x^4$, but we do not use these functions for the reasons explained in Section 6.

In this special case, the approximation in (6) simplifies to

$$H(X) \approx H(\nu) - [k_1(E\{\bar{G}_1(X)\})^2 + k_2(E\{\bar{G}_2(X)\} - E\{\bar{G}_2(\nu)\})^2] \qquad (7)$$

where $k_1$ and $k_2$ are positive constants (see [8]), and $\nu$ is a Gaussian random variable of zero mean and unit variance. Practical examples of choices of $\bar{G}_i$ that are consistent with the requirements in Section 6 are the following.

First, for measuring bimodality/sparsity, one might use, according to the recommendations of Section 6, the log-density of the double exponential (or Laplace) distribution: $\bar{G}_{2a}(x) = |x|$. For computational reasons, a smoother version of $\bar{G}_{2a}$ might also be used. Another choice would be the Gaussian function, which may be considered as the log-density of a distribution with infinitely heavy tails: $\bar{G}_{2b}(x) = \exp(-x^2/2)$. For measuring asymmetry, one might use, on more heuristic grounds, the following function: $\bar{G}_1(x) = x\exp(-x^2/2)$. which corresponds to the second term in the projection pursuit index proposed in [3].

Using the above examples one obtains two practical examples of (7):

$$H_a(X) = H(\nu) - [k_1(E\{X\exp(-X^2/2)\})^2 + k_2^a(E\{|X|\} - \sqrt{2/\pi})^2], \qquad (8)$$

$$H_b(X) = H(\nu) - [k_1(E\{X\exp(-X^2/2)\})^2 + k_2^b(E\{\exp(-X^2/2)\} - \sqrt{1/2})^2], \qquad (9)$$

with $k_1 = 36/(8\sqrt{3} - 9)$, $k_2^a = 1/(2 - 6/\pi)$, and $k_2^b = 24/(16\sqrt{3} - 27)$. As above, $H(\nu) = \frac{1}{2}(1 + \log(2\pi))$ means the entropy of a standardized Gaussian variable. These approximations $H_a(X)$ and $H_b(X)$ can be considered more robust and accurate generalizations of the approximation derived using the Gram-Charlier expansion in [9]. Indeed, using the polynomials $\bar{G}_1(x) = x^3$ and $\bar{G}_2(x) = x^4$ one obtains the approximation of entropy in [9], which is in practice almost identical to those proposed in [1, 2]. Finally, note that the approximation in (9) is very similar to the first two terms of the projection pursuit index in [3]. Algorithms for independent component analysis and projection pursuit can be derived from these approximations, see [7].

## 8   Simulation results

To show the validity of our approximations of differential entropy we compared the approximations $H_a$ and $H_b$ in Eqs (8) and (9) in Section 7, with the one offered by higher-order cumulants as given in [9]. The expectations were here evaluated exactly, ignoring finite-sample effects.

First, we used a family of Gaussian mixture densities, defined by

$$f(x) = \mu\varphi(x) + (1 - \mu)2\varphi(2(x - 1))  \tag{10}$$

where $\mu$ is a parameter that takes all the values in the interval $0 \le \mu \le 1$. This family includes asymmetric densities of both negative and positive kurtosis. The results are depicted in Fig. 1. Note that the plots show approximations of negentropies: the negentropy of $X$ equals $H(\nu) - H(X)$, where $\nu$ is again a standardized Gaussian variable. One can see that both of the approximations $H_a$ and $H_b$ introduced in Section 7 were considerably more accurate than the cumulant-based approximation.

Second, we considered the following family of density functions:

$$f_\alpha(x) = C_1 \exp(C_2|x|^\alpha)  \tag{11}$$

where $\alpha$ is a positive constant, and $C_1, C_2$ are normalization constants that make $f_\alpha$ a probability density of unit variance. For different values of $\alpha$, the densities in this family exhibit different shapes. For $\alpha < 2$, one obtains (sparse) densities of positive kurtosis. For $\alpha = 2$, one obtains the Gaussian density, and for $\alpha > 2$, a density of negative kurtosis. Thus the densities in this family can be used as examples of different symmetric non-Gaussian densities. In Figure 2, the different approximations are plotted for this family, using parameter values $.5 \le \alpha \le 3$. Since the densities used are all symmetric, the first terms in the approximations were neglected. Again, it is clear that both of the approximations $H_a$ and $H_b$ introduced in Section 7 were much more accurate than the cumulant-based approximation in [2, 9]. (In the case of symmetric densities, these two cumulant-based approximations are identical). Especially in the case of sparse densities (or densities of positive kurtosis), the cumulant-based approximations performed very poorly; this is probably because it gives too much weight to the tails of the distribution.

## References

[1] S. Amari, A. Cichocki, and H.H. Yang. A new learning algorithm for blind source separation. In D. S. Touretzky, M. C. Mozer, and M. E. Hasselmo, editors, *Advances in Neural Information Processing 8 (Proc. NIPS'95)*, pages 757–763. MIT Press, Cambridge, MA, 1996.

[2] P. Comon. Independent component analysis – a new concept? *Signal Processing*, 36:287–314, 1994.

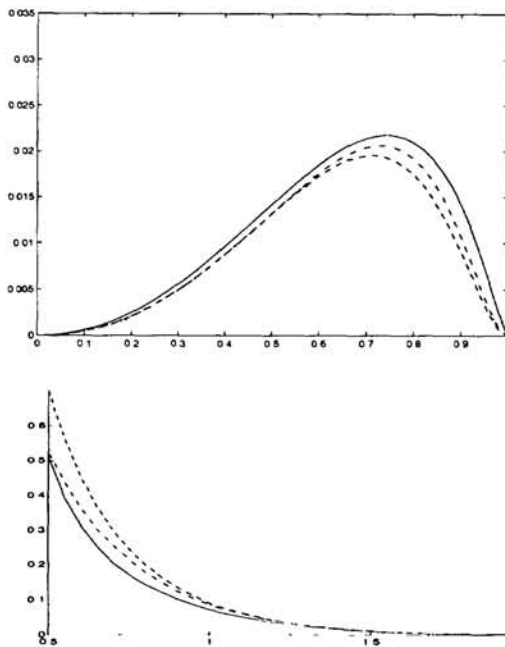

Figure 1: Comparison of different approximations of negentropy, for the family of mixture densities in (10) parametrized by $\mu$ ranging from 0 to 1. Solid curve: true negentropy. Dotted curve: cumulant-based approximation. Dashed curve: approximation $H_a$ in (8). Dot-dashed curve: approximation $H_b$ in (9). Our two approximations were clearly better than the cumulant-based one.

Figure 2: Comparison of different approximations of negentropy, for the family of densities (11) parametrized by $\alpha$. On the left, approximations for densities of positive kurtosis ($.5 \le \alpha < 2$) are depicted, and on the right, approximations for densities of negative kurtosis ($2 < \alpha \le 3$). Solid curve: true negentropy. Dotted curve: cumulant-based approximation. Dashed curve: approximation $H_a$ in (8). Dot-dashed curve: approximation $H_b$ in (9). Clearly, our two approximations were much better than the cumulant-based one, especially in the case of densities of positive kurtosis.

[3] D. Cook, A. Buja, and J. Cabrera. Projection pursuit indexes based on orthonormal function expansions. *J. of Computational and Graphical Statistics*, 2(3):225–250, 1993.

[4] T. M. Cover and J. A. Thomas. *Elements of Information Theory*. John Wiley & Sons, 1991.

[5] J.H. Friedman. Exploratory projection pursuit. *J. of the American Statistical Association*, 82(397):249–266, 1987.

[6] P.J. Huber. Projection pursuit. *The Annals of Statistics*, 13(2):435–475, 1985.

[7] A. Hyvärinen. Independent component analysis by minimization of mutual information. Technical Report A46, Helsinki University of Technology, Laboratory of Computer and Information Science, 1997.

[8] A. Hyvärinen. New approximations of differential entropy for independent component analysis and projection pursuit. Technical Report A47, Helsinki University of Technology, Laboratory of Computer and Information Science, 1997. Available at http://www.cis.hut.fi/~aapo.

[9] M.C. Jones and R. Sibson. What is projection pursuit ? *J. of the Royal Statistical Society, ser. A*, 150:1–36, 1987.

[10] C. Jutten and J. Herault. Blind separation of sources, part I: An adaptive algorithm based on neuromimetic architecture. *Signal Processing*, 24:1–10, 1991.

[11] M. Kendall and A. Stuart. *The Advanced Theory of Statistics*. Charles Griffin & Company, 1958.
